# Surface Learning with Applications to Lipreading

**Christoph Bregler** *,**
*Computer Science Division
University of California
Berkeley, CA 94720

**Stephen M. Omohundro** **
**Int. Computer Science Institute
1947 Center Street Suite 600
Berkeley, CA 94704

## Abstract

Most connectionist research has focused on learning mappings from one space to another (eg. classification and regression). This paper introduces the more general task of learning constraint surfaces. It describes a simple but powerful architecture for learning and manipulating nonlinear surfaces from data. We demonstrate the technique on low dimensional synthetic surfaces and compare it to nearest neighbor approaches. We then show its utility in learning the space of lip images in a system for improving speech recognition by lip reading. This learned surface is used to improve the visual tracking performance during recognition.

## 1 Surface Learning

Mappings are an appropriate representation for systems whose variables naturally decompose into "inputs" and "outputs". To use a learned mapping, the input variables must be known and error-free and a single output value must be estimated for each input. Many tasks in vision, robotics, and control must maintain relationships between variables which don't naturally decompose in this way. Instead, there is a nonlinear constraint surface on which the values of the variables are jointly restricted to lie. We propose a representation for such surfaces which supports a wide range of queries and which can be naturally learned from data.

The simplest queries are "completion queries". In these queries, the values of certain variables are specified and the values (or constraints on the values) of remaining

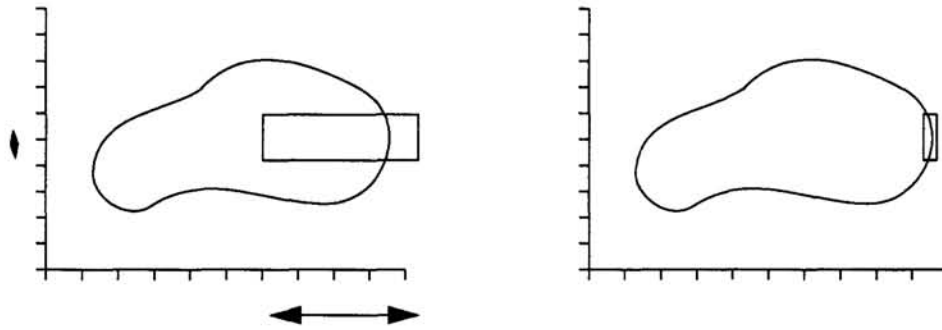

Figure 1: Using a constraint surface to reduce uncertainty in two variables

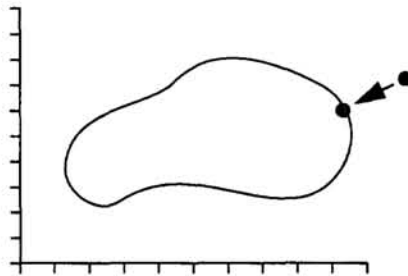

Figure 2: Finding the closest point in a surface to a given point.

variables are to be determined. This reduces to a conventional mapping query if the "input" variables are specified and the system reports the values of corresponding "output" variables. Such queries can also be used to invert mappings, however, by specifying the "output" variables in the query. Figure 1 shows a generalization in which the variables are known to lie with certain ranges and the constraint surface is used to further restrict these ranges.

For recognition tasks, "nearest point" queries in which the system must return the surface point which is closest to a specified sample point are important (Figure 2). For example, symmetry-invariant classification can be performed by taking the surface to be generated by applying all symmetry operations to class prototypes (eg. translations, rotations, and scalings of exemplar characters in an OCR system). In our representation we are able to efficiently find the globally nearest surface point in this kind of query.

Other important classes of queries are "interpolation queries" and "prediction queries". For these, two or more points on a curve are specified and the goal is to interpolate between them or extrapolate beyond them. Knowledge of the constraint surface can dramatically improve performance over "knowledge-free" approaches like linear or spline interpolation.

In addition to supporting these and other queries, one would like a representation which can be efficiently learned. The training data is a set of points randomly drawn from the surface. The system should generalize from these training points to form a representation of the surface (Figure 3). This task is more difficult than mapping learning for several reasons: 1) The system must discover the dimension of the surface, 2) The surface may be topologically complex (eg. a torus or a sphere)

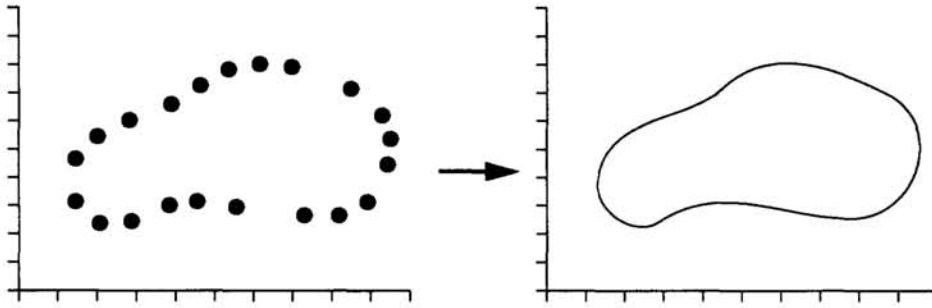

Figure 3: Surface Learning

and may not support a single set of coordinates, 3) The broader range of queries discussed above must be supported.

Our approach starts from the observation that if the data points were drawn from a *linear* surface, then a principle components analysis could be used to discover the dimension of the linear space and to find the best-fit linear space of that dimension. The largest principle vectors would span the space and there would be a precipitous drop in the principle values at the dimension of the surface. A principle components analysis will no longer work, however, when the surface is nonlinear because even a 1-dimensional curve could be embedded so as to span all the dimensions of the space.

If a nonlinear surface is smooth, however, then each local piece looks more and more linear under magnification. If we consider only those data points which lie within a local region, then to a good approximation they come from a linear surface patch. The principle values can be used to determine the most likely dimension of the surface and that number of the largest principle components span its tangent space (Omohundro, 1988). The key idea behind our representations is to "glue" these local patches together using a partition of unity.

We are exploring several implementations, but all the results reported here come from a represenation based on the "nearest point" query. The surface is represented as a mapping from the embedding space to itself which takes each point to the nearest surface point. K-means clustering is used to determine a initial set of "prototype centers" from the data points. A principle components analysis is performed on a specified number of the nearest neighbors of each prototype. These "local PCA" results are used to estimate the dimension of the surface and to find the best linear projection in the neighborhood of prototype $i$. The influence of these local models is determined by Gaussians centered on the prototype location with a variance determined by the local sample density. The projection onto the surface is determined by forming a partition of unity from these Gaussians and using it to form a convex linear combination of the local linear projections:

$$P(x) = \frac{\sum_i G_i(x) P_i(x)}{\sum_i G_i(x)} \qquad (1)$$

This initial model is then refined to minimize the mean squared error between the

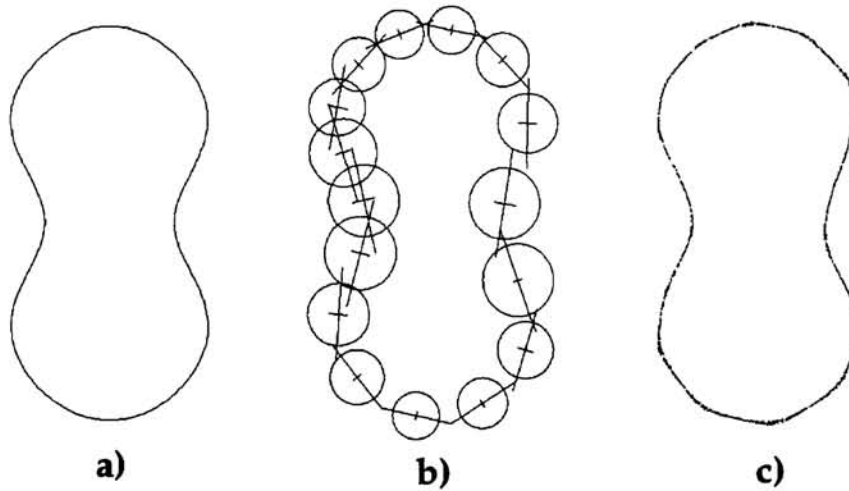

Figure 4: Learning a 1-dimensional surface. a) The surface to learn b) The local patches and the range of their influence functions, c) The learned surface

training samples and the nearest surface point using EM optimization and gradient descent.

## 2    Synthetic Examples

To see how this approach works, consider 200 samples drawn from a 1-dimensional curve in a two-dimensional space (Figure 4a). 16 prototype centers are chosen by k-means clustering. At each center, a local principle components analysis is performed on the closest 20 training samples. Figure 4b shows the prototype centers and the two local principle components as straight lines. In this case, the larger principle value is several times larger than the smaller one. The system therefore attempts to construct a one-dimensional learned surface. The circles in Figure 4b show the extent of the Gaussian influence functions for each prototype. Figure 4c shows the resulting learned suface. It was generated by randomly selecting 2000 points in the neighborhood of the surface and projecting them according to the learned model.

Figure 5 shows the same process applied to learning a two-dimensional surface embedded in three dimensions.

To quantify the performance of this learning algorithm, we studied the effect of the different parameters on learning a two-dimensional sphere in three dimensions. It is easy to compare the learned results with the correct ones in this case. Figure 6a shows how the empirical error in the nearest point query decreases as a function of the number of training samples. We compare it against the error made by a nearest-neighbor algorithm. With 50 training samples our approach produces an error which is one-fourth as large. Figure 6b shows how the average size of the local principle values depends on the number of nearest neighbors included. Because this is a two-dimensional surface, the two largest values are well-separated from the third largest. The rate of growth of the principle values is useful for determining the dimension of the surface in the presence of noise.

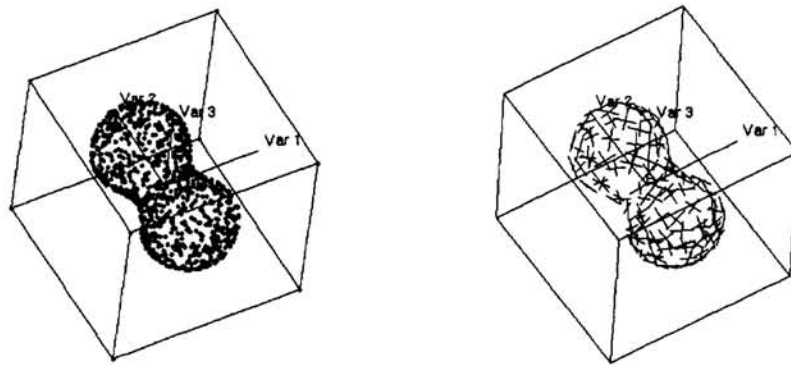

Figure 5: Learning a two-dimensional surface in the three dimensions a) 1000 random samples on the surface b) The two largest local principle components at each of 100 prototype centers based on 25 nearest neighbors.

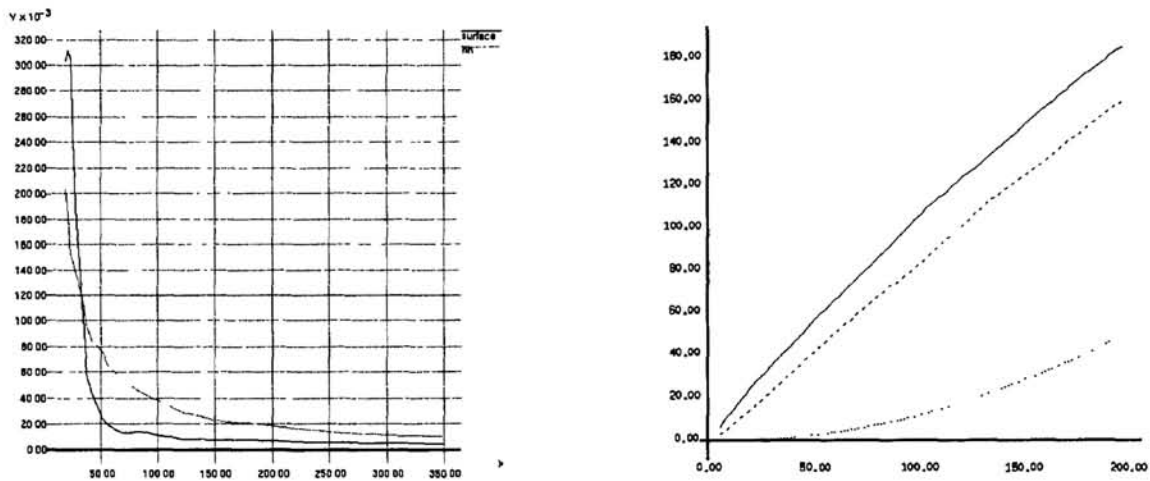

Figure 6: Quantitative performance on learning a two-dimensional sphere in three dimensions. a) Mean squared error of closest point querries as function of the number of samples for the learned surface vs. nearest training point b) The mean square root of the three principle values as a function of number of neighbors included in each local PCA.

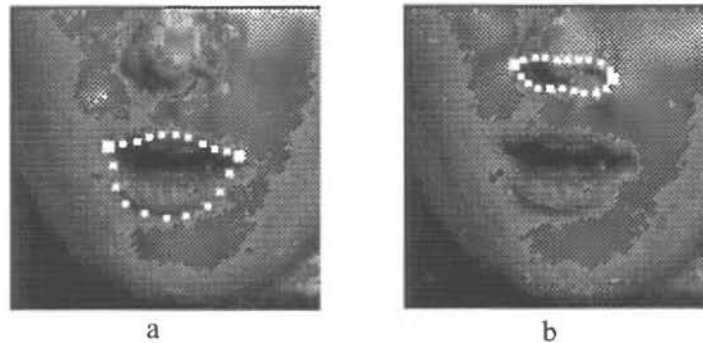

a                                    b

Figure 7: Snakes for finding the lip contours a) A correctly placed snake b) A snake which has gotten stuck in a local minimum of the simple energy function.

## 3    Modelling the space of lips

We are using this technique as a part of system to do "lipreading". To provide features for "viseme classification" (visemes are the visual analog of phonemes), we would like the system to reliably track the shape of a speaker's lips in video images. It should be able to identify the corners of the lips and to estimate the bounding curves robustly under a variety of imaging and lighting conditions. Two approaches to this kind of tracking task are "snakes" (Kass, et. al, 1987) and "deformable templates" (Yuille, 1991). Both of these approaches minimize an "energy function" which is a sum of an internal model energy and an energy measuring the match to external image features.

For example, to use the "snake" approach for lip tracking, we form the internal energy from the first and second derivatives of the coordinates along the snake, preferring smoother snakes to less smooth ones. The external energy is formed from an estimate of the negative image gradient along the snake. Figure 7a shows a snake which has correctly relaxed onto a lip contour. This energy function is not very specific to lips, however. For example, the internal energy just causes the snake to be a controlled continuity spline. The "lip- snakes" sometimes relax onto undesirable local minima like that shown in Figure 7b. Models based on deformable templates allow a researcher to more strongly constrain the shape space (typically with hand-coded quadratic linking polynomials), but are difficult to use for representing fine grain lip features.

Our approach is to use surface learning as described here to build a model of the space of lips. We can then replace the internal energy described above by a quantity computed from the distance to the learned surface in lip feature space.

Our training set consists of 4500 images of a speaker uttering random words[1]. The training images are initially "labeled" with the conventional snake algorithm. Incorrectly aligned snakes are removed from the database by hand. The contour shape is parameterized by the $x$ and $y$ coordinates of 40 evenly spaced points along the snake. All values are normalized to give a lip width of 1. Each lip contour is

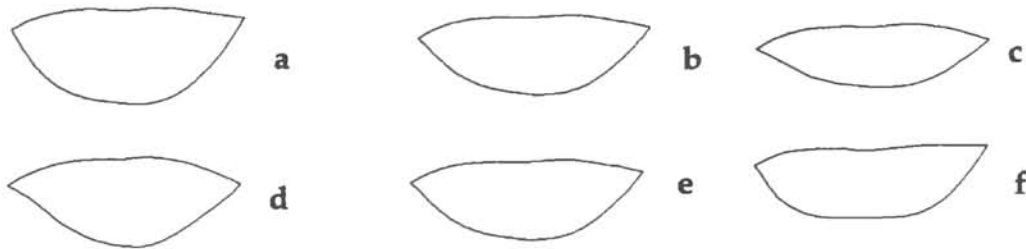

Figure 8: Two principle axes in a local patch in lip space. a, b, and c are configurations along the first principle axis, while d, e, and f are along the third axis.

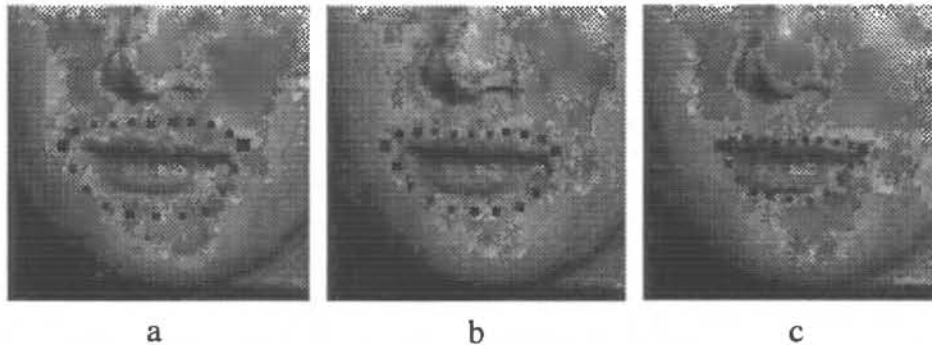

Figure 9: a) Initial crude estimate of the contour b) An intermediate step in the relaxation c) The final contour.

therefore a point in an 80-dimensional "lip- space". The lip configurations which actually occur lie on a lower dimensional surface embedded in this space. Our experiments show that a 5-dimensional surface in the 80-dimensional lip space is sufficient to describe the contours with single pixel accuracy in the image. Figure 8 shows some lip models along two of the principle axes in the local neighborhood of one of the patches. The lip recognition system uses this learned surface to improve the performance of tracking on new image sequences.

The tracking algorithm starts with a crude initial estimate of the lip position and size. It chooses the closest model in the lip surface and maps the corresponding resized contour back onto the estimated image position (Figure 9a). The external image energy is taken to be the cumulative magnitude of graylevel gradient estimates along the current contour. This term has maximum value when the curve is aligned exactly on the lip boundary. We perform gradient ascent in the contour space, but constrain the contour to lie in the learned lip surface. This is achieved by reprojecting the contour onto the lip surface after each gradient step. The surface thereby acts as the analog of the internal energy in the snake and deformable template approaches. Figure 9b shows the result after a few steps and figure 9c shows the final contour. The image gradient is estimated using an image filter whose width is gradually reduced as the search proceeds.

The lip contours in successive images in the video sequence are found by starting with the relaxed contour from the previous image and performing gradient ascent

with the altered external image energies. Empirically, surface-based tracking is far more robust than the "knowledge-free" approaches. While we have described the approach in the context of contour finding, it is much more general and we are currently extending the system to model more complex aspects of the image.

The full lipreading system which combines the described tracking algorithm and a hybrid connectionist speech recognizer (MLP/HMM) is described in (Bregler and Konig 1994). Additionally we will use the lip surface to interpolate visual features to match them with the higher rate auditory features.

## 4   Conclusions

We have presented the task of learning surfaces from data and described several important queries that the learned surfaces should support: completion, nearest point, interpolation, and prediction. We have described an algorithm which is capable of efficiently performing these tasks and demonstrated it on both synthetic data and on a real-world lip-tracking problem. The approach can be made computationally efficient using the "bumptree" data structure described in (Omohundro, 1991). We are currently studying the use of "model merging" to improve the representation and are also applying it to robot control.

**Acknowledgements**

This research was funded in part by Advanced Research Project Agency contract #N0000 1493 C0249 and by the International Computer Science Institute. The database was collected with a grant from Land Baden Wuerttenberg (Landesschwerpunkt Neuroinformatik) at Alex Waibel's institute.

**References**

C. Bregler, H. Hild, S. Manke & A. Waibel. (1993) Improving Connected Letter Recognition by Lipreading. In *Proc. of Int. Conf. on Acoustics, Speech, and Signal Processing, Minneapolis.*

C. Bregler, Y. Konig (1994) "Eigenlips" for Robust Speech Recognition. In *Proc. of Int. Conf. on Acoustics, Speech, and Signal Processing, Adelaide.*

M. Kass, A. Witkin, and D. Terzopoulos. (1987) SNAKES: Active Contour Models, in *Proc. of the First Int. Conf. on Computer Vision, London.*

S. Omohundro. (1988) Fundamentals of Geometric Learning. University of Illinois at Urbana-Champaign Technical Report UIUCDCS-R-88-1408.

S. Omohundro. (1991) Bumptrees for Efficient Function, Constraint, and Classification Learning. In Lippmann, Moody, and Touretzky (ed.), *Advances in Neural Information Processing Systems 3.* San Mateo, CA: Morgan Kaufmann.

A. Yuille. (1991) Deformable Templates for Face Recognition, *Journal of Cognitive Neuroscience*, Volume 3, Number 1.

## Footnotes

[1]The data was collected for an earlier lipreading system described in (Bregler, Hild, Manke, Waibel 1993)
